# Probability Estimation from a Database Using a Gibbs Energy Model

**John W. Miller**
Microsoft Research (9/1051)
One Microsoft Way
Redmond, WA 98052

**Rodney M. Goodman**
Dept. of Electrical Engineering (116-81)
California Institute of Technology
Pasadena, CA 91125

## Abstract

We present an algorithm for creating a neural network which produces accurate probability estimates as outputs. The network implements a Gibbs probability distribution model of the training database. This model is created by a new transformation relating the joint probabilities of attributes in the database to the weights (Gibbs potentials) of the distributed network model. The theory of this transformation is presented together with experimental results. One advantage of this approach is the network weights are prescribed without iterative gradient descent. Used as a classifier the network tied or outperformed published results on a variety of databases.

## 1 INTRODUCTION

This paper addresses the problem of modeling a discrete database. The database is viewed as a collection of independent samples from a probability distribution. This distribution is called the *underlying* distribution. In contrast, the empirical distribution is the distribution obtained if you take independent random samples from the database (with replacement). The task of creating a probability model can be separated into two parts. The first part is the problem of choosing statistics of the samples which are expected to accurately represent the underlying distribution. The second part is the problem of choosing a model which is consistent with these statistics. Under reasonable assumptions, the optimal solution to the second problem is the method of *Maximum Entropy*. For a broad class of statistics, the

Maximum Entropy solution is a *Gibbs* probability distribution (Slepian, 1972). In this paper, the background and theoretical result of a transformation from joint statistics to a Gibbs energy (or network weight) representation is presented. We then outline the experimental test results of an efficient algorithm implementing this transform without using gradient descent iteration.

## 2    BACKGROUND

Define a set $T$ to be the set of attributes (or fields) in a database. For a particular entry (or record) of the database, define the associated set of attribute values to be the configuration $\omega$ of the attributes. The set of attribute values associated with a subset $b \subset T$ is called a subconfiguration $\omega_b$. Using this set notation the Gibbs probability distribution may be defined:

$$p(\omega) = Z^{-1} \cdot e^{V_T(\omega)} \tag{1}$$

where

$$V_T(\omega) = \sum_{b \subseteq T} J_b(\omega) \tag{2}$$

The function $V$ is called the *energy*. The function $J_b$, called the *potential function*, defines a real value for every subconfiguration of the set $b$. $Z$ is the normalizing constant that makes the sum of probabilities of all configurations equal to unity.

Prior work in the neural network literature using the Gibbs distribution (such as the Boltzmann Machine) has primarily used second order models ($J_b = 0$ if $|b| > 2$) (Hinton, 1986). By adding new attributes not in the original database, second order potentials have been used to model complex distributions. The work presented in this paper, in contrast, uses higher order potentials to model complex probability distributions. We begin by considering the case where every potential of every order is used to model the distribution.

The Principle of Inclusion-Exclusion from set theory states that the following two equations are equivalent:

$$g(A) = \sum_{b \subseteq A} f(b) \tag{3}$$

$$f(A) = \sum_{b \subseteq A} (-1)^{|A-b|} g(b). \tag{4}$$

The method of inverting an equation from the form of (3) into one in the form of (4) is a special case of *Möbius Inversion*. Clifford-Hammersley (Kindermann, 1980) used this relation to invert formula (2):

$$J_A(\omega) = \sum_{b \subseteq A} (-1)^{|A-b|} V_b(\omega) \tag{5}$$

Define the probability of a subconfiguration $p(\omega_b)$ to be the probability that the attributes in set $b$ take on the values defined in the configuration $\omega$. Using (1) to describe the probability distribution of subconfigurations, equation (5) can be written:

$$J_A(\omega) = \sum_{b \subseteq A} (-1)^{|A-b|} \ln(\, p(\omega_b)\,) \tag{6}$$

## 3  A TRANSFORMATION TO GIBBS POTENTIALS

Equation (6) provides a technique for modeling distributions by potential functions rather than directly through the observable joint statistics of sets of attributes. If the model is truncated by setting high order potentials to zero, then the energy model becomes an estimate of the model obtained by collecting the joint statistics, rather than an exact equivalent. If equation (6) is used directly, the error in the energy due to setting all potentials of order $d$ to zero grows quickly with $d$. For this reason (6) must be normalized if it is going to be used in a truncated modeling scheme. A normalization version of equation (2) that corrects for the unequal number of potentials of different orders is:

$$V_A(\omega) = \sum_{b \subseteq A} \binom{|A| - 1}{|b| - 1}^{-1} J_b(\omega) \qquad (7)$$

This equation can be inverted to show the surprising result, a weight associated with $\omega_A$:

$$J_A(\omega) = \ln(p_A(\omega)) - (|A| - 1)^{-1} \sum_{\substack{t \in A \\ b = A - t}} \ln(p_b(\omega)) \qquad (8)$$

For example, with three attribute values $\{x, y, z\}$, the following potentials are defined:

$$J_{\{x\}} = \ln(p(x))$$
$$J_{\{y\}} = \ln(p(y))$$
$$J_{\{z\}} = \ln(p(z))$$
$$J_{\{xy\}} = \ln\left(\frac{p(xy)}{p(x)p(y)}\right)$$
$$J_{\{yz\}} = \ln\left(\frac{p(yz)}{p(y)p(z)}\right)$$
$$J_{\{xz\}} = \ln\left(\frac{p(xz)}{p(x)p(z)}\right)$$
$$J_{\{xyz\}} = \ln\left(\frac{p(xyz)}{\sqrt{p(xy)p(yz)p(xz)}}\right)$$

For a given database sample, a potential is *activated* if all of its defined attribute values are true for the sample. The weighted sum of all activated potentials recovers an approximation of the probability of the database sample. If all potentials of every order have been used to create the model, then this approximation is exactly the probability of the sample in the empirical distribution. The correct weighting is given by equation (7). For example it is easily verified that:

$$\ln(p(xyz)) = \binom{2}{2}^{-1} J_{\{xyz\}} + \binom{2}{1}^{-1} (J_{\{xy\}} + J_{\{xz\}} + J_{\{yz\}})$$
$$+ \binom{2}{0}^{-1} (J_{\{x\}} + J_{\{y\}} + J_{\{z\}}).$$

The Gibbs model truncated to second order potentials would estimate the probability in this example by:

$$\ln(p(xyz)) \approx \binom{2}{1}^{-1}(J_{\{xy\}} + J_{\{xz\}} + J_{\{yz\}}) + \binom{2}{0}^{-1}(J_{\{x\}} + J_{\{y\}} + J_{\{z\}}).$$

$$\approx \ln\sqrt{p(xy)p(yz)p(xz)}$$

## 4    PROOF OF THE INVERSION FORMULA

**Theorem:**
Let $T$ be a finite set. Each element of $T$ will be called an attribute. Each attribute can take on one of a finite set of states called attribute values. A collection of attribute values for every element of $T$ is called a configuration $\omega$. For all $A \subseteq T$ (including both the empty set $A = \emptyset$ and the full set $A = T$), let $V_A(\omega)$ and $J_A(\omega)$ be functions mapping the states of the elements of $A$ to the real numbers. Define $\binom{m}{n} = m!/((m-n)! \cdot n!)$ to be "m choose n."
Let $V_\emptyset(\omega) = 0$, $J_\emptyset(\omega) = 0$, and let $V_A(\omega) = J_A(\omega)$ if $|A| = 1$.
Then for $|A| > 1$:

$$V_A(\omega) = \sum_{b \subseteq A} \binom{|A|-1}{|b|-1}^{-1} J_b(\omega), \tag{9}$$

and

$$J_A(\omega) = V_A(\omega) - \sum_{\substack{b \subset A \\ |b|=|A|-1}} (|A|-1)^{-1} \cdot V_b(\omega) \tag{10}$$

are equivalent in that any assignment of $V_A$ and $J_A$ values for all $A \subseteq T$ will satisfy (9) if and only if they also satisfy (10).

**Proof:**
Let $\mathcal{J}$ be any assignment of the values $J_A(\omega)$ for all $A \subseteq T$. Let $\mathcal{V}$ be any assignment of all the values $V_A(\omega)$ for all $A \subseteq T$. Then clearly (9) maps any assignment $\mathcal{J}$ to a unique $\mathcal{V}$. We will represent this mapping by the function $f$, so (9) is abbreviated $\mathcal{V} = f(\mathcal{J})$. Similarly (10) maps any assignment $\mathcal{V}$ to a unique $\mathcal{J}$. Equation (10) will be abbreviated $\mathcal{J} = g(\mathcal{V})$. The result of Lemma C1 below, applied with the value $\mathcal{D}$ set to $n$, shows that $f(g(\mathcal{V})) = \mathcal{V}$. In Lemma C2 below, it is shown $g(f(\mathcal{J})) = \mathcal{J}$. Therefore the equations (9) and (10) are inverse one-to-one mappings and the association of assignments between $\mathcal{J}$ and $\mathcal{V}$ are identical for the two equations.                                                             *Q.E.D.*

**Lemma C1:**
Rather than simply showing $f(g(\mathcal{V})) = \mathcal{V}$, a more general result will be shown. Since the number of potentials of a given order increases exponentially with the order, it is useful to approximate the energy of a configuration by defining a maximum order $\mathcal{D}$ such that all potentials of greater order are assumed to be zero

$$J_b(\omega) = 0 \quad \forall \; b \text{ such that } |b| > \mathcal{D}.$$

Let $\hat{V}_A(\omega)$ be the resulting approximation to the energy $V_A(\omega)$. Let $|A| = n$.

Given

$$J_A(\omega) = V_A(\omega) - \sum_{\substack{b \subset A \\ |b|=n-1}} (n-1)^{-1} \cdot V_b(\omega) \tag{11}$$

and the order $\mathcal{D}$ approximation to equation (7):

$$\hat{V}_A(\omega) = \sum_{i=1}^{\mathcal{D}} \binom{n-1}{i-1}^{-1} \sum_{\substack{b \subseteq A \\ |b|=i}} J_b(\omega), \tag{12}$$

then

$$\hat{V}_A(\omega) = \sum_{\substack{b \subseteq A \\ |b|=\mathcal{D}}} \binom{n-1}{\mathcal{D}-1}^{-1} V_b(\omega).$$

**Note:**
For the case $\mathcal{D} = n$, the approximation is exact

$$\hat{V}_A(\omega) = V_A(\omega).$$

and so $f(g(\mathcal{V})) = \mathcal{V}$ is shown.

The lemma's result has a simple interpretation. The energy of a configuration is approximated by a scaled average of the energies of the configurations of order $\mathcal{D}$. Using equation (1) to relate energies to probabilities, shows that the estimated probability is a scaled geometric mean of the order $\mathcal{D}$ marginal probabilities.

**Proof:**
We start with the given equation for $\hat{V}_A(\omega)$

$$\hat{V}_A(\omega) = \sum_{i=1}^{\mathcal{D}} \binom{n-1}{i-1}^{-1} \sum_{\substack{b \subseteq A \\ |b|=i}} J_b(\omega).$$

Use equation (11) to substitute $J_b(\omega)$ out of the equation:

$$\hat{V}_A(\omega) = \sum_{i=1}^{\mathcal{D}} \binom{n-1}{i-1}^{-1} \sum_{\substack{b \subseteq A \\ |b|=i}} \left( V_b(\omega) - \sum_{\substack{c \subset b, |b| \neq 1 \\ |c|=|b|-1}} (i-1)^{-1} \cdot V_c(\omega) \right)$$

Separate the term in the first sum where $i = \mathcal{D}$

$$\hat{V}_A(\omega) = \sum_{\substack{b \subseteq A \\ |b|=\mathcal{D}}} \binom{n-1}{\mathcal{D}-1}^{-1} V_b(\omega) + \left( \sum_{i=1}^{n-1} \binom{n-1}{i-1}^{-1} \sum_{\substack{b \subseteq A \\ |b|=i}} V_b(\omega) \right)$$
$$- \sum_{i=1}^{n} \binom{n-1}{i-1}^{-1} \sum_{\substack{b \subseteq A \\ |b|=i}} \sum_{\substack{c \subset b, |b| \neq 1 \\ |c|=|b|-1}} (i-1)^{-1} \cdot V_c(\omega).$$

By subtracting $\hat{V}_A(\omega)$ from both sides using equation (12) and noting the second summation over $i$ has no terms when $i = 1$ we see that it is sufficient to show

$$\sum_{i=1}^{\mathcal{D}-1} \binom{n-1}{i-1}^{-1} \sum_{\substack{b \subseteq A \\ |b|=i}} V_b(\omega) = \sum_{i=2}^{\mathcal{D}} \binom{n-1}{i-1}^{-1} \sum_{\substack{b \subseteq A \\ |b|=i}} \sum_{\substack{c \subset b \\ |c|=|b|-1}} (i-1)^{-1} \cdot V_c(\omega).$$

The right hand side inner double summation counts a given $V_c(\omega)$ once for every $b$ such that $c \subset b \subseteq A$ with $i = |b| = |c| + 1$. This occurs exactly $|A| - |c| = n - i + 1$ times. Thus

$$\sum_{i=1}^{D-1} \binom{n-1}{i-1}^{-1} \sum_{\substack{b \subset A \\ |b|=i}} V_b(\omega) = \sum_{i=2}^{D} \binom{n-1}{i-1}^{-1} \sum_{\substack{c \subset A \\ |c|=i-1}} \frac{n-i+1}{i-1} \cdot V_c(\omega).$$

Now perform a change of variables. Let $j = i - 1$ on the right hand side

$$\sum_{i=1}^{D-1} \binom{n-1}{i-1}^{-1} \sum_{\substack{b \subset A \\ |b|=i}} V_b(\omega) = \sum_{j=1}^{D-1} \binom{n-1}{j}^{-1} \sum_{\substack{c \subset A \\ |c|=j}} \frac{n-j}{j} \cdot V_c(\omega).$$

Clearly both sides are identical since

$$\binom{n-1}{i-1}^{-1} = \binom{n-1}{i}^{-1} \frac{n-i}{i}.$$

<div align="right">Q.E.D.</div>

**Lemma C2:** $g(f(\mathcal{J})) = \mathcal{J}$

Let $|A| = n$. It is sufficient to show that substituting $V_b$ out of (10) using (9) yields an identity:

$$J_A(\omega) = V_A(\omega) - \sum_{\substack{b \subset A, n \neq 1 \\ |b|=n-1}} (n-1)^{-1} \cdot V_b(\omega)$$

$$= \sum_{b \subseteq A} \binom{n-1}{|b|-1}^{-1} J_b(\omega) - \sum_{\substack{b \subset A, n \neq 1 \\ |b|=n-1}} (n-1)^{-1} \sum_{c \subseteq b} \binom{|b|-1}{|c|-1}^{-1} J_c(\omega).$$

Separate the term in the first sum for which $b = A$

$$J_A(\omega) = J_A(\omega)$$
$$+ \sum_{\substack{b \subset A \\ b \neq A}} \binom{n-1}{|b|-1}^{-1} J_b(\omega) - \sum_{\substack{b \subset A, n \neq 1 \\ |b|=n-1}} (n-1)^{-1} \sum_{c \subseteq b} \binom{|b|-1}{|c|-1}^{-1} J_c(\omega).$$

Subtract $J_A(\omega)$ from both sides. The right hand side double sum counts a given $J_c(\omega)$ once for every $b$ such that $c \subseteq b \subset A$ with $|b| = |A| - 1 = n - 1$. This occurs $|A| - |c| = n - |c|$ times. It is sufficient to show

$$\sum_{b \subset A, b \neq A} \binom{n-1}{|b|-1}^{-1} J_b(\omega) = \sum_{c \subset A, c \neq A} \frac{n-|c|}{n-1} \binom{n-2}{|c|-1}^{-1} J_c(\omega).$$

Both sides are identical since:

$$\binom{n-1}{i-1}^{-1} = \frac{n-i}{n-1} \binom{n-2}{i-1}^{-1}.$$

<div align="right">Q.E.D.</div>

# 5   USING THE INVERSION FORMULA TO SET NETWORK WEIGHTS

Our method of probability estimation is to first collect empirical frequencies of patterns (subconfigurations) from the database. (An efficient hash table implementation of the algorithm is described in (Miller, 1993). The basic idea is to remove from the database a pattern with low potential whenever there is a hash collision which prevents a new pattern count from being stored.) Second, interpreting these frequencies as probabilities, we convert each pattern frequency to a potential using equation (8). We assume patterns with unknown or uncalculated frequencies have zero potential. Low order patterns which never occur are assigned a large negative potential (this approximation is needed to model events with zero probability in the empirical distribution). Finally, we calculate the probability of any new pattern not in the training set using the neural network implementation of equations (7) and (1).

# 6   RESULTS

One way to validate the performance of a probability model is to test its performance as a classifier. The probability model is used as a classifier by calculating the probabilities of each unknown class value together with the known attribute values. The most probable combination is then chosen as the predicted class. Used as a classifier the Gibbs model tied or outperformed published results on a variety of databases. Table 1 outlines results on three datasets taken from the UC Irvine archive (Murphy, 1992). The Gibbs model results were collected from the very first experiment using the algorithm with the datasets. No difficult parameter adjustment is necessary to get the algorithm to classify at these rates. The iris database has 4 real value attributes. Each attribute was quantized into a decile ranking for use by the algorithm.

# 7   CONCLUSION

A new method of extracting a Gibbs probability model from a database has been presented. The approach uses the Principle of Inclusion-Exclusion to invert a set of collected statistics into a set of potentials for a Gibbs energy model. A hash table implementation is used to efficiently process database records in order to collect the most important potentials, or weights, which can be stored in the available memory. Although the model is designed to give accurate probability estimates rather than simply class labels, the model in practice works well as a classifier on a variety of databases.

**Acknowledgements**

This work is funded in part by DARPA and ONR under grant N00014-92-J-1860.

Table 1: Summary of Classification Results

| Database | A | C | R | Train | Test | Trials | Gibbs Rate | Compare |
|---|---|---|---|---|---|---|---|---|
| House Voting | 16 | 2 | 435 | 335 | 100 | 50 | 95.3% | 95% |
| Iris | 4 | 3 | 150 | 120 | 30 | 100 | 96.3% | n.a. |
| Iris | 4 | 3 | 150 | 149 | 1 | 1000 | 97.1% | 98.0% |
| Breast Cancer | 9 | 2 | 699 | 599 | 100 | 100 | 97.3% | n.a. |
| Breast Cancer | 9 | 2 | 369 | 200 | 169 | 100 | 95.7% | 93.7% |

A = Attribute count in the database, excluding the class attribute
C = Class count
R = Record count
Train = Number of records used to create the energy for one trial
Test = Number of records tested in a single trial
Trials = Number of independent train-test trials used to calculate the rate
Gibbs Rate = Gibbs energy model classification rate
Compare = Baseline classification result of other methods (Schlimmer, 1987), (Weiss, 1992),(Zhang, 1992) respectively

## References

D. Slepian, "On Maxentropic Discrete Stationary Processes," *Bell System Technical Journal*, **51**, pp.629–653, 1972.

G.E. Hinton and T.J. Sejnowski, "Learning and Relearning in Boltzmann Machines," in *Parallel Distributed Processing*, Vol. I., pp.282–317, Cambridge MA: MIT Press, 1986.

R. Kindermann, J.L. Snell, *Markov Random Fields and their Applications*, Providence, RI: American Mathematical Society, 1980.

J. W. Miller, "Building Probabilistic Models from Databases" California Institute of Technology, Ph.D. Thesis 1993.

P. Murphy, and D. Aha, *UCI Repository of Machine Learning Databases* [Machine-readable data repository at ics.uci.edu in directory /pub/machine-learning-databases]. Irvine, CA: University of California, Department of Information and Computer Science, 1992.

Schlimmer, J. C., "Concept Acquisition Through Representational Adjustment" University of California at Irvine, Ph.D. Thesis 1987.

S. Weiss, and I. Kapouleas, "An Empirical Comparison of Pattern Recognition, Neural Nets, and Machine Learning Classification Methods," in *Proceedings of the 11th International Joint Conference on Artificial Intelligence* Vol. 1, pp.781–787, Los Gatos, CA: Morgan Kaufmann, 1992.

J. Zhang, "Selecting Typical Instances in Instance-Based Learning," in *Proceedings of the Ninth International Machine Learning Conference* Aberdeen, Scotland, pp.470–479, San Mateo CA: Morgan Kaufmann, 1992.
